# Kernels for gene regulatory regions

**Jean-Philippe Vert**
Geostatistics Center
Ecole des Mines de Paris - ParisTech
Jean-Philippe.Vert@ensmp.fr

**Robert Thurman**
Division of Medical Genetics
University of Washington
rthurman@u.washington.edu

**William Stafford Noble**
Department of Genome Sciences
University of Washington
noble@gs.washington.edu

## Abstract

We describe a hierarchy of motif-based kernels for multiple alignments of biological sequences, particularly suitable to process regulatory regions of genes. The kernels incorporate progressively more information, with the most complex kernel accounting for a multiple alignment of orthologous regions, the phylogenetic tree relating the species, and the prior knowledge that relevant sequence patterns occur in conserved motif blocks. These kernels can be used in the presence of a library of known transcription factor binding sites, or *de novo* by iterating over all $k$-mers of a given length. In the latter mode, a discriminative classifier built from such a kernel not only recognizes a given class of promoter regions, but as a side effect simultaneously identifies a collection of relevant, discriminative sequence motifs. We demonstrate the utility of the motif-based multiple alignment kernels by using a collection of aligned promoter regions from five yeast species to recognize classes of cell-cycle regulated genes. Supplementary data is available at http://noble.gs.washington.edu/proj/pkernel.

## 1 Introduction

In a eukaryotic cell, a variety of DNA switches—promoters, enhancers, silencers, etc.—regulate the production of proteins from DNA. These switches typically contain multiple binding site motifs, each of length 5–15 nucleotides, for a class of DNA-binding proteins known as transcription factors. As a result, the detection of such regulatory motifs proximal to a gene provides important clues about its regulation and, therefore, its function. These motifs, if known, are consequently interesting features to extract from genomic sequences in order to compare genes, or cluster them into functional families.

These regulatory motifs, however, usually represent a tiny fraction of the intergenic sequence, and their automatic detection remains extremely challenging. For well-studied transcription factors, libraries of known binding site motifs can be used to scan the intergenic sequence. A common approach for the *de novo* detection of regulatory motifs is to

start from a set of genes known to be similarly regulated, for example by clustering gene expression data, and search for over-represented short sequences in their proximal intergenic regions. Alternatively, some authors have proposed to represent each intergenic sequence by its content in short sequences, and to correlate this representation with gene expression data [1]. Finally, additional information to characterize regulatory motifs can be gained by comparing the intergenic sequences of orthologous genes, i.e., genes from different species that have evolved from a common ancestor, because regulatory motifs are more conserved than non-functional intergenic DNA [2].

We propose in this paper a hierarchy of increasingly complex representations for intergenic sequences. Each representation yields a positive definite kernel between intergenic sequences. While various motif-based sequence kernels have been described in the literature (e.g., [3, 4, 5]), these kernels typically operate on sequences from a single species, ignoring relevant information from orthologous sequences. In contrast, our hierarchy of motif-based kernels accounts for a multiple alignment of orthologous regions, the phylogenetic tree relating the species, and the prior knowledge that relevant sequence patterns occur in conserved motif blocks. These kernels can be used in the presence of a library of known transcription factor binding sites, or *de novo* by iterating over all $k$-mers of a given length. In the latter mode, a discriminative classifier built from such a kernel not only recognizes a given class of regulatory sequences, but as a side effect simultaneously identifies a collection of discriminative sequence motifs. We demonstrate the utility of the motif-based multiple alignment kernels by using a collection of aligned intergenic regions from five yeast species to recognize classes of co-regulated genes.

From a methodological point of view, this paper can be seen as an attempt to incorporate an increasing amount of prior knowledge into a kernel. In particular, this prior information takes the form of a probabilistic model describing with increasing accuracy the object we want to represent. All kernels were designed before any experiment was conducted, and we then performed an objective empirical evaluation of each kernel without further parameter optimization. In general, classification performance improved as the amount of prior knowledge increased. This observation supports the notion that tuning a kernel with prior knowledge is beneficial. However, we observed no improvement in performance following the last modification of the kernel, highlighting the fact that a richer model of the data does not always lead to better performance accuracy.

## 2   Kernels for intergenic sequences

In a complex eukaryotic genome, regulatory switches may occur anywhere within a relatively large genomic region near a given gene. In this work we focus on a well-studied model organism, the budding yeast *Saccharomyces cerevisiae*, in which the typical intergenic region is less than 1000 bases long. We refer to the intergenic region upstream of a yeast gene as its *promoter region*. Denoting the four-letter set of nucleotides as $\mathcal{A} = \{A, C, G, T\}$, the promoter region of a gene is a finite-length sequence of nucleotides $x \in \mathcal{A}^* = \bigcup_{i=0}^{\infty} \mathcal{A}^i$. Given several sequenced organisms, *in silico* comparison of genes between organisms often allows the detection of orthologous genes, that is, genes that evolved from a common ancestor. If the species are evolutionarily close, as are different yeast strains, then the promoter regions are usually quite similar and can be represented as a *multiple alignment*. Each position in this alignment represents one letter in the shared ancestor's promoter region. Mathematically speaking, a multiple alignment of length $n$ of $p$ sequences is a sequence $\mathbf{c} = c_1, c_2, \ldots, c_n$, where each $c_i \in \bar{\mathcal{A}}^p$, for $i = 1, \ldots, n$, is a column of $p$ letters in the alphabet $\bar{\mathcal{A}} = \mathcal{A} \cup \{-\}$. The additional letter "$-$" is used to represent gaps in sequences, which represent insertion or deletion of letters during the evolution of the sequences.

We are now in the position to describe a family of representations and kernels for promoter

regions, incorporating an increasing amount of prior knowledge about the properties of regulatory motifs. All kernels below are simple inner products between vector representations of promoter regions. These vector representations are always indexed by a set $\mathcal{M}$ of short sequences of fixed length $d$, which can either be all $d$-mers, i.e., $\mathcal{M} = \mathcal{A}^d$, or a predefined library of indexing sequences. A promoter region $P$ (either single sequence or multiple alignment) is therefore always represented by a vector $\Phi_{\mathcal{M}}(P) = (\Phi_a(P))_{a \in \mathcal{M}}$.

**Motif kernel on a single sequence**   The simplest approach to index a *single* promoter region $\mathbf{x} \in \mathcal{A}^*$ with an alphabet $\mathcal{M}$ is to define

$$\Phi_a^{\text{Spectrum}}(\mathbf{x}) = n_a(\mathbf{x}) , \quad \forall a \in \mathcal{M} ,$$

where $n_a(\mathbf{x})$ counts the number of occurrences of $a$ in $\mathbf{x}$. When $\mathcal{M} = \mathcal{A}^d$, the resulting kernel is the spectrum kernel [3] between single promoter regions.

**Motif kernel on multiple sequences**   When a gene has $p$ orthologs in other species, then a set of $p$ promoter regions $\{\mathbf{x}_1, \mathbf{x}_2, \ldots, \mathbf{x}_p\} \in (\mathcal{A}^*)^p$, which are expected to contain similar regulatory motifs, is available. We propose the following representation for such a set:

$$\Phi_a^{\text{Summation}}(\{\mathbf{x}_1, \mathbf{x}_2, \ldots, \mathbf{x}_p\}) = \sum_{i=1}^{p} \Phi_a^{\text{Spectrum}}(\mathbf{x}_i) , \quad \forall a \in \mathcal{M} .$$

We call the resulting kernel the *summation* kernel. It is essentially the spectrum kernel on the concatenation of the available promoter regions—ignoring, however, $k$-mers that overlap different sequences in the concatenation. The rationale behind this kernel, compared to the spectrum kernel, is two-fold. First, if all promoters contain common functional motifs and randomly varying nonfunctional motifs, then the signal-to-noise ratio of the relevant regulatory features compared to other irrelevant non-functional features increases by taking the sum (or mean) of individual feature vectors. Second, even functional motifs representing transcription factor binding sites are known to have some variability in some positions, and merging the occurrences of a similar motif in different sequences is a way to model this flexibility in the framework of a vector representation.

**Marginalized motif kernel on a multiple alignment**   The summation kernel might suffer from at least two limitations. First, it does not include any information about the relationships between orthologs, in particular their relative similarities. Suppose for example that three species are compared, two of them being very similar. Then the promoter regions of two out of three orthologs would be virtually identical, giving an unjustified double weight to this duplicated species compared to the third one in the summation kernel. Second, although mutations in functional motifs between different species would correspond to different short motifs in the summation kernel feature vector, these varying short motifs might not cover all allowed variations in the functional motifs, especially if the motifs are extracted from a small number of orthologs. In such cases, probabilistic models such as weight matrices, which estimate possible variations for each position independently, are known to make more efficient use of the data.

In order to overcome these limitations, we propose to transform the set of promoter regions into a multiple alignment. We therefore assume that a fixed number of $q$ species has been selected, and that a probabilistic model $p(h, c)$, with $h \in \bar{\mathcal{A}}$ and $c \in \bar{\mathcal{A}}^q$ has been tuned on these species. By "tuned," we mean that $p(h, c)$ is a distribution that accurately describes the probability of a given letter $h$ in the common ancestor of the species, together with the set of letters $c$ at the corresponding position in the set of species. Such distributions are commonly used in computational genomics, often resulting from the estimation of a phylogenetic tree model [6]. We also assume that all sets of $q$ promoter regions of groups of orthologous genes in the $q$ species have been turned into multiple alignments.

Given an alignment $\mathbf{c} = c_1, c_2, \ldots, c_n$, suppose for the moment that we know the corresponding true sequence of nucleotides of the common ancestor $\mathbf{h} = h_1, h_2, \ldots, h_n$. Then the spectrum of the sequence $\mathbf{h}$, that is, $\Phi_{\mathcal{M}}^{\text{Spectrum}}(\mathbf{h})$, would be a good summary for the multiple alignment, and the inner product between two such spectra would be a candidate kernel between multiple alignments. The sequence $\mathbf{h}$ being of course unknown, we propose to estimate its conditional probability given the multiple alignment $\mathbf{c}$, under the model where all columns are independent and identically distributed according to the evolutionary model, that is, $p(\mathbf{h}|\mathbf{c}) = \prod_{i=1}^{n} p(h_i|c_i)$. Under this probabilistic model, it is now possible to define the representation of the multiple alignment as the expectation of the spectrum representation of $\mathbf{h}$ with respect to this conditional probability, that is:

$$\Phi_a^{\text{Marginalized}}(\mathbf{c}) = \sum_{\mathbf{h}} \Phi_a^{\text{Spectrum}}(\mathbf{h})p(\mathbf{h}|\mathbf{c}) , \quad \forall a \in \mathcal{M} . \tag{1}$$

In order to compute this representation, we observe that if $\mathbf{h}$ has length $n$ and $a = a_1 \ldots a_d$ has length $d$, then

$$\Phi_a^{\text{Spectrum}}(\mathbf{h}) = \sum_{i=1}^{n-d+1} \delta(a, h_i \ldots h_{i+d-1}) ,$$

where $\delta$ is the Kronecker function. Therefore,

$$\begin{aligned}
\Phi_a^{\text{Marginalized}}(\mathbf{c}) &= \sum_{\mathbf{h} \in \mathcal{A}^n} \left\{ \left( \sum_{i=1}^{n-d+1} \delta(a, h_i \ldots h_{i+d-1}) \right) \prod_{i=1}^{n} p(h_i|c_i) \right\} \\
&= \sum_{i=1}^{n-d+1} \left( \prod_{j=0}^{d-1} p(a_{j+1}|c_{i+j}) \right) .
\end{aligned}$$

This computation can be performed explicitly by computing $p(a_{j+1}|c_{i+j})$ at each position $i = 1, \ldots, n$, and performing the sum of the products of these probabilities over a moving window. We call the resulting kernel the *marginalized* kernel because it corresponds to the marginalization of the spectrum kernel under the phylogenetic probabilistic model [7].

**Marginalized motif kernel with phylogenetic shadowing**   The marginalized kernel is expected to be useful when relevant information is distributed along the entire length of the sequences analyzed. In the case of promoter regions, however, the relevant information is more likely to be located within a few short motifs. Because only a small fraction of the total set of promoter regions lies within such motifs, this information is likely to be lost when the whole sequence is represented by its spectrum. In order to overcome this limitation, we exploit the observation that relevant motifs are more evolutionarily conserved on average than the surrounding sequence. This hypothesis has been confirmed by many studies that show that functional parts, being under more evolutionary pressure, are more conserved than non-functional ones.

Given a multiple alignment $c$, let us assume (temporarily) that we know which parts are relevant. We can encode this information into a sequence of binary variables $\mathbf{s} = s_1 \ldots s_n \in \{0,1\}^n$, where $s_i = 1$ means that the $i$th position is relevant, and irrelevant if $s_i = 0$. A typical sequence for a promoter region consist primarily of 0's, except for a few positions indicating the position of the transcription factor binding motifs. Let us also assume that we know the nucleotide sequence $\mathbf{h}$ of the common ancestor. Then it would make sense to use a spectrum kernel based on the spectrum of $\mathbf{h}$ restricted to the relevant positions only. In other words, all positions where $s_i = 0$ could be thrown away, in order to focus only on the relevant positions. This corresponds to defining the features:

$$\Phi_a^{\text{Relevant}}(\mathbf{h}, \mathbf{s}) = \sum_{i=1}^{n-d+1} \delta(a, h_i \ldots h_{i+d-1})\delta(s_i, 1) \ldots \delta(s_{i+d-1}, 1) , \quad \forall a \in \mathcal{M} .$$

Given only a multiple alignment $\mathbf{c}$, the sequences $\mathbf{h}$ and $\mathbf{s}$ are not known but can be estimated. This is where the hypothesis that relevant nucleotides are more conserved than irrelevant nucleotides can be encoded, by using two models of evolution with different rates of mutations, as in phylogenetic shadowing [2]. Let us therefore assume that we have a model $p(c|h, s = 0)$ that describes "fast" evolution from an ancestral nucleotide $h$ to a column $c$ in a multiple alignment, and a second model $p_1(c|h, s = 1)$ that describes "slow" evolution. In practice, we take these models to be two classical evolutionary models with different mutation rates, but any reasonable pair of random models could be used here, if one had a better model for functional sites, for example. Given these two models of evolution, let us also define a prior probability $p(s)$ that a position is relevant or not (related to the proportion of relevant parts we expect in a promoter region), and prior probabilities for the ancestor nucleotide $p(h|s = 0)$ and $p(h|s = 1)$.

The joint probability of being in state $s$, having an ancestor nucleotide $h$ and a resulting alignment $c$ is then $p(c, h, s) = p(s)p(h|s)p(c|h, s)$. Under the probabilistic model where all columns are independent from each other, that is, $p(\mathbf{h}, \mathbf{s}|\mathbf{c}) = \prod_{i=1}^{n} p(h_i, s_i|c_i)$, we can now replace (1) by the following features:

$$\Phi_a^{\text{Shadow}}(\mathbf{c}) = \sum_{\mathbf{h}, \mathbf{s}} \Phi_a^{\text{Relevant}}(\mathbf{h}, \mathbf{s})p(\mathbf{h}, \mathbf{s}|\mathbf{c}) , \quad \forall a \in \mathcal{M} . \tag{2}$$

Like the marginalized spectrum kernel, this kernel can be computed by computing the explicit representation of each multiple sequence alignment $\mathbf{c}$ as a vector $(\Phi_a(\mathbf{c}))_{a \in \mathcal{M}}$ as follows:

$$\Phi_a^{\text{Shadow}}(\mathbf{c}) = \sum_{\mathbf{h} \in \mathcal{A}^n} \sum_{\mathbf{s} \in \{0,1\}^n} \left\{ \sum_{i=1}^{n-d+1} \delta(a, h_i \dots h_{i+d-1})\delta(s_i, 1) \dots \delta(s_{i+d-1}, 1) \prod_{i=1}^{n} p(h_i, s_i|c_i) \right\}$$

$$= \sum_{i=1}^{n-d+1} \left( \prod_{j=0}^{d-1} p(h = a_{j+1}, s = 1|c_{i+j}) \right) .$$

The computation can then be carried out by exploiting the observation that each term can be computed by:

$$p(h, s = 1|c) = \frac{p(s = 1)p(h|s = 1)p(c|h, s = 1)}{p(s = 0)p(c|s = 0) + p(s = 1)p(c|s = 1)}.$$

Moreover, it can easily be seen that, like the marginalized kernel, the shadow kernel is the marginalization of the kernel corresponding to $\Phi^{\text{Relevant}}$ with respect to $p(\mathbf{h}, \mathbf{s}|\mathbf{c})$.

**Incorporating Markov dependencies between positions**  The probabilistic model used in the shadow kernel models each position independently from the others. As a result, a conserved position has the same contribution to the shadow kernel if it is surrounded by other conserved positions, or by varying positions. In order to encode our prior knowledge that the pattern of functional / nonfunctional positions along the sequence is likely to be a succession of short functional regions and longer nonfunctional regions, we propose to replace this probabilistic model by a probabilistic model with a Markov dependency between successive positions for the variable $\mathbf{s}$, that is, to consider the probability:

$$p^{\text{Markov}}(\mathbf{c}, \mathbf{h}, \mathbf{s}) = p(s_1)p(h_1, c_1|s_1) \prod_{i=2}^{n} p(s_i|s_{i-1}) p(h_i, c_i|s_i).$$

This suggests replacing (2) by

$$\Phi_a^{\text{Markov}}(\mathbf{c}) = \sum_{\mathbf{h}, \mathbf{s}} \Phi_a(\mathbf{h}, \mathbf{s})p^{\text{Markov}}(\mathbf{h}, \mathbf{s}|\mathbf{c}) , \quad \forall a \in \mathcal{M} .$$

Once again, this feature vector can be computed as a sum of window weights over sequences by

$$\Phi_a^{\text{Markov}}(\mathbf{c}) = \sum_{i=1}^{n-d+1} \Big( p\,(s_i = 1|\mathbf{c})\, p\,(h_i = a_{j+1}|c_i, s_i = 1)$$
$$\times \prod_{j=1}^{d-1} p(h_{i+j} = a_{j+1}, s_{i+j} = 1|c_{i+j}, s_{i+j-1} = 1) \Big)\,.$$

The main difference with the computation of the shadow kernel is the need to compute the term $P\,(s_i = 1|\mathbf{c})$, which can be done using the general sum-product algorithm.

## 3   Experiments

We measure the utility of our hierarchy of kernels in a cross-validated, supervised learning framework. As a starting point for the analysis, we use various groups of genes that show co-expression in a microarray study. Eight gene groups were derived from a study that applied hierarchical clustering to a collection of 79 experimental conditions, including time series from the diauxic shift, the cell cycle series, and sporulation, as well as various temperature and reducing shocks [8]. We hypothesize that co-expression implies co-regulation of a given group of genes by a common set of transcription factors. Hence, the corresponding promoter regions should be enriched for a corresponding set of transcription factor binding motifs. We test the ability of a support vector machine (SVM) classifier to learn to recapitulate the co-expression classes, based only upon the promoter regions. Our results show that the SVM performance improves as we incorporate more prior knowledge into the promoter kernel.

We collected the promoter regions from five closely related yeast species [9, 10]. Promoter regions from orthologous genes were aligned using ClustalW, discarding promoter regions that aligned with less than 30% sequence identity relative to the other sequences in the alignment. This procedure produced 3591 promoter region alignments. For the phylogenetic kernels, we inferred a phylogenetic tree among the five yeast species from alignments of four highly conserved proteins—MCM2, MCM3, CDC47 and MCM6. The concatenated alignment was analyzed with fastDNAml [11] using the default parameters. The resulting tree was used in all of our analyses.

SVMs were trained using Gist (microarray.cpmc.columbia.edu/gist) with the default parameters. These include a normalized kernel, and a two-norm soft margin with asymmetric penalty based upon the ratio of positive and negative class sizes. All kernels were computed by summing over all $4^5$ $k$-mers of width 5. Each class was recognized in a one-vs-all fashion. SVM testing was performed using balanced three-fold cross-validation, repeated five times.

The results of this experiment are summarized in Table 1. For every gene class, the worst-performing kernel is one of the three simplest kernels: "simple," "summation" or "marginalization." The mean ROC scores across all eight classes for these three kernels are 0.733, 0.765 and 0.748. Classification performance improves dramatically using the shadow kernel with either a small (2) or large (5) ratio of fast-to-slow evolutionary rates. The mean ROC scores for these two kernels are 0.854 and 0.844. Furthermore, across five of the eight gene classes, one of the two shadow kernels is the best-performing kernel. The Markov kernel performs approximately as well as the shadow kernel. We tried six different parameterizations, as shown in the table, and these achieved mean ROC scores ranging from 0.822 to 0.850. The differences between the best parameterization of this kernel ("Markov 5 90/90") and "shadow 2" are not significant. Although further tuning

Table 1: **Mean ROC scores for SVMs trained using various kernels to recognize classes of co-expressed yeast genes.** The second row in the table gives the number of genes in each class. All other rows contain mean ROC scores across balanced three-fold cross-validation, repeated five times. Standard errors (not shown) are almost uniformly 0.02, with a few values of 0.03. Values in bold-face are the best mean ROC for the given class of genes. The classes of genes (columns) are, respectively, ATP synthesis, DNA replication, glycolysis, mitochondrial ribosome, proteasome, spindle-pole body, splicing and TCA cycle. The kernels are as described in the text. For the shadow and Markov kernels, the values "2" and "5" refer to the ratio of fast to slow evolutionary rates. For the Markov kernel, the values "90" and "99" refer to the self-transition probabilities (times 100) in the conserved and varying states of the model.

| Kernel | ATP 15 | DNA 5 | Glyc 17 | Ribo 22 | Prot 27 | Spin 11 | Splic 14 | TCA 16 | Mean |
|---|---|---|---|---|---|---|---|---|---|
| single | 0.711 | 0.777 | 0.814 | 0.743 | 0.735 | 0.716 | 0.683 | 0.684 | 0.733 |
| summation | 0.773 | 0.768 | 0.824 | 0.750 | 0.763 | 0.756 | 0.739 | 0.740 | 0.764 |
| marginalized | 0.799 | 0.805 | 0.833 | 0.729 | 0.748 | 0.721 | 0.676 | 0.673 | 0.748 |
| shadow 2 | 0.881 | 0.929 | **0.928** | 0.840 | 0.867 | **0.827** | 0.787 | **0.770** | **0.854** |
| shadow 5 | **0.889** | **0.935** | 0.927 | 0.819 | 0.849 | 0.821 | 0.766 | 0.752 | 0.845 |
| Markov 2 90/90 | 0.848 | 0.891 | 0.908 | 0.830 | 0.853 | 0.801 | 0.773 | 0.758 | 0.833 |
| Markov 2 90/99 | 0.868 | 0.911 | 0.915 | 0.826 | 0.850 | 0.782 | 0.752 | 0.735 | 0.830 |
| Markov 2 99/99 | 0.869 | 0.910 | 0.912 | 0.816 | 0.840 | 0.773 | 0.737 | 0.724 | 0.823 |
| Markov 5 90/90 | 0.875 | 0.922 | 0.924 | **0.844** | **0.868** | 0.814 | **0.788** | 0.769 | 0.851 |
| Markov 5 90/99 | 0.872 | 0.916 | 0.920 | 0.834 | 0.858 | 0.794 | 0.774 | 0.755 | 0.840 |
| Markov 5 99/99 | 0.868 | 0.917 | 0.921 | 0.830 | 0.853 | 0.774 | 0.751 | 0.733 | 0.831 |

of kernel parameters might yield significant improvement, our results thus far suggest that incorporating dependencies between adjacent positions does not help very much.

Finally, we test the ability of the SVM to identify sequence regions that correspond to biologically significant motifs. As a gold standard, we use the JASPAR database (jaspar.cgb.ki.se), searching each class of promoter regions using MONKEY (rana.lbl.gov/~alan/Monkey.htm) with a $p$-value threshold of $10^{-4}$. For each gene class, we identify the three JASPAR motifs that occur most frequently within that class, and we create a list of all 5-mers that appear within those motif occurrences. Next, we create a corresponding list of 5-mers identified by the SVM. We do this by calculating the hyperplane weight associated with each 5-mer and retaining the top 20 5-mers for each of the 15 cross-validation runs. We then take the union over all runs to come up with a list of between 40 and 55 top 5-mers for each class. Table 2 indicates that the discriminative 5-mers identified by the SVM are significantly enriched in 5-mers that appear within biologically significant motif regions, with significant $p$-values for all eight gene classes (see caption for details).

## 4   Conclusion

We have described and demonstrated the utility of a class of kernels for characterizing gene regulatory regions. These kernels allow us to incorporate prior knowledge about the evolution of a set of orthologous sequences and the conservation of transcription factor binding site motifs. We have also demonstrated that the motifs identified by an SVM trained using these kernels correspond to biologically significant motif regions. Our future work will focus on automating the process of agglomerating the identified $k$-mers into a smaller set of motif models, and on applying these kernels in combination with gene expression, protein-protein interaction and other genome-wide data sets.

This work was funded by NIH awards R33 HG003070 and U01 HG003161.

Table 2: **SVM features correlate with discriminative motifs.** The first row lists the number of non-redundant 5-mers constructed from high-scoring SVM features. Row two gives the number of 5-mers constructed from JASPAR motif occurrences in the 5-species alignments. Row three is a tally of all 5-mers appearing in the sequences making up the class. The fourth row gives the size of the intersection between the SVM and motif-based 5-mer lists. The final two rows give the expected value and p-value for the intersection size. The p-value is computed using the hypergeometric distribution by enumerating all possibilites for the intersection of two sets selected from a larger set given the sizes in the first three rows.

|         | ATP     | DNA     | Glyc    | Ribo     | Prot    | Spin    | Splic   | TCA     |
|---------|---------|---------|---------|----------|---------|---------|---------|---------|
| SVM     | 46      | 40      | 55      | 50       | 49      | 43      | 48      | 50      |
| Motif   | 180     | 68      | 227     | 38       | 148     | 152     | 52      | 104     |
| Class   | 1006    | 839     | 967     | 973      | 1001    | 891     | 881     | 995     |
| Inter   | 24      | 8       | 23      | 18       | 23      | 19      | 14      | 21      |
| Expect  | 8.23    | 3.24    | 12.91   | 1.95     | 7.25    | 7.34    | 2.83    | 5.23    |
| $p$-value | 6.19e-8 | 1.15e-2 | 1.44e-3 | 3.88e-15 | 3.24e-8 | 1.74e-5 | 1.15e-7 | 2.00e-9 |

# References

[1] D. Y. Chiang, P. O. Brown, and M. B. Eisen. Visualizing associations between genome sequences and gene expression data using genome-mean expression profiles. *Bioinformatics*, 17(Supp. 1):S49–S55, 2001.

[2] D. Boffelli, J. McAuliffe, D. Ovcharenko, K. D. Lewis, I. Ovcharenko, L. Pachter, and E. M. Rubin. Phylogenetic shadowing of primate sequences to find functional regions of the human genome. *Science*, 299:1391–1394, 2003.

[3] C. Leslie, E. Eskin, and W. S. Noble. The spectrum kernel: A string kernel for SVM protein classification. In R. B. Altman, A. K. Dunker, L. Hunter, K. Lauderdale, and T. E. Klein, editors, *Proceedings of the Pacific Symposium on Biocomputing*, pages 564–575, New Jersey, 2002. World Scientific.

[4] X. H-F. Zhang, K. A. Heller, I. Hefter, C. S. Leslie, and L. A. Chasin. Sequence information for the splicing of human pre-mRNA identified by support vector machine classification. *Genome Research*, 13:2637–2650, 2003.

[5] A. Zien, G. Rätch, S. Mika, B. Schölkopf, T. Lengauer, and K.-R. Müller. Engineering support vector machine kernels that recognize translation initiation sites. *Bioinformatics*, 16(9):799–807, 2000.

[6] R. Durbin, S. Eddy, A. Krogh, and G. Mitchison. *Biological Sequence Analysis*. Cambridge UP, 1998.

[7] K. Tsuda, T. Kin, and K. Asai. Marginalized kernels for biological sequences. *Bioinformatics*, 18:S268–S275, 2002.

[8] M. Eisen, P. Spellman, P. O. Brown, and D. Botstein. Cluster analysis and display of genome-wide expression patterns. *Proceedings of the National Academy of Sciences of the United States of America*, 95:14863–14868, 1998.

[9] Paul Cliften, Priya Sudarsanam, Ashwin Desikan, Lucinda Fulton, Bob Fulton, John Majors, Robert Waterston, Barak A. Cohen, and Mark Johnston. Finding functional features in Saccharomyces genomes by phylogenetic footprinting. *Science*, 301(5629):71–76, 2003.

[10] Manolis Kellis, Nick Patterson, Matthew Endrizzi, Bruce Birren, and Eric S Lander. Sequencing and comparison of yeast species to identify genes and regulatory elements. *Nature*, 423(6937):241–254, 2003.

[11] GJ Olsen, H Matsuda, R Hagstrom, and R Overbeek. fastDNAmL: a tool for construction of phylogenetic trees of DNA sequences using maximum likelihood. *Comput. Appl. Biosci.*, 10(1):41–48, 1994.
